# Clustering data through an analogy to the Potts model

**Marcelo Blatt, Shai Wiseman and Eytan Domany**
Department of Physics of Complex Systems,
The Weizmann Institute of Science, Rehovot 76100, Israel

## Abstract

A new approach for clustering is proposed. This method is based
on an analogy to a physical model; the ferromagnetic Potts model
at thermal equilibrium is used as an analog computer for this hard
optimization problem. We do not assume any structure of the un-
derlying distribution of the data. Phase space of the Potts model is
divided into three regions; ferromagnetic, super-paramagnetic and
paramagnetic phases. The region of interest is that corresponding
to the super-paramagnetic one, where domains of aligned spins ap-
pear. The range of temperatures where these structures are stable
is indicated by a non-vanishing magnetic susceptibility. We use a
very efficient Monte Carlo algorithm to measure the susceptibil-
ity and the spin spin correlation function. The values of the spin
spin correlation function, at the super-paramagnetic phase, serve
to identify the partition of the data points into clusters.

Many natural phenomena can be viewed as optimization processes, and the drive to
understand and analyze them yielded powerful mathematical methods. Thus when
wishing to solve a hard optimization problem, it may be advantageous to apply these
methods through a physical analogy. Indeed, recently techniques from statistical
physics have been adapted for solving hard optimization problems (see *e.g.* Yuille
and Kosowsky, 1994). In this work we formulate the problem of clustering in terms
of a ferromagnetic Potts spin model. Using the Monte Carlo method we estimate
physical quantities such as the spin spin correlation function and the susceptibility,
and deduce from them the number of clusters and cluster sizes.
Cluster analysis is an important technique in exploratory data analysis and is ap-
plied in a variety of engineering and scientific disciplines. The problem of *partitional
clustering* can be formally stated as follows. With every one of $i = 1, 2, \ldots N$ pat-
terns represented as a point $\vec{x}_i$ in a $d$-dimensional metric space, determine the
partition of these $N$ points into $M$ groups, called *clusters*, such that points in a
cluster are more similar to each other than to points in different clusters. The value
of $M$ also has to be determined.

The two main approaches to partitional clustering are called *parametric* and *non-parametric*. In parametric approaches some knowledge of the clusters' structure is assumed (*e.g.* each cluster can be represented by a center and a spread around it). This assumption is incorporated in a *global criterion*. The goal is to assign the data points so that the criterion is minimized. A typical example is *variance minimization* (Rose, Gurewitz, and Fox, 1993). On the other hand, in non-parametric approaches a *local criterion* is used to build clusters by utilizing local structure of the data. For example, clusters can be formed by identifying high-density regions in the data space or by assigning a point and its $K$-nearest neighbors to the same cluster. In recent years many parametric partitional clustering algorithms rooted in statistical physics were presented (see *e.g.* Buhmann and Kühnel , 1993). In the present work we use methods of statistical physics in non-parametric clustering.

Our aim is to use a physical problem as an analog to the clustering problem. The notion of clusters comes very naturally in Potts spin models (Wang and Swendsen, 1990) where clusters are closely related to ordered regions of spins. We place a Potts spin variable $s_i$ at each point $\vec{x}_i$ (that represents one of the patterns), and introduce a short range ferromagnetic interaction $J_{ij}$ between pairs of spins, whose strength decreases as the inter-spin distance $\|\vec{x}_i - \vec{x}_j\|$ increases. The system is governed by the Hamiltonian (energy function)

$$\mathcal{H} = - \sum_{<i,j>} J_{ij} \delta_{s_i, s_j} \qquad s_i = 1 \ldots q , \qquad (1)$$

where the notation $< i, j >$ stands for neighboring points $i$ and $j$ in a sense that is defined later. Then we study the ordering properties of this inhomogeneous Potts model.

As a concrete example, place a Potts spin at each of the data points of fig. 1.

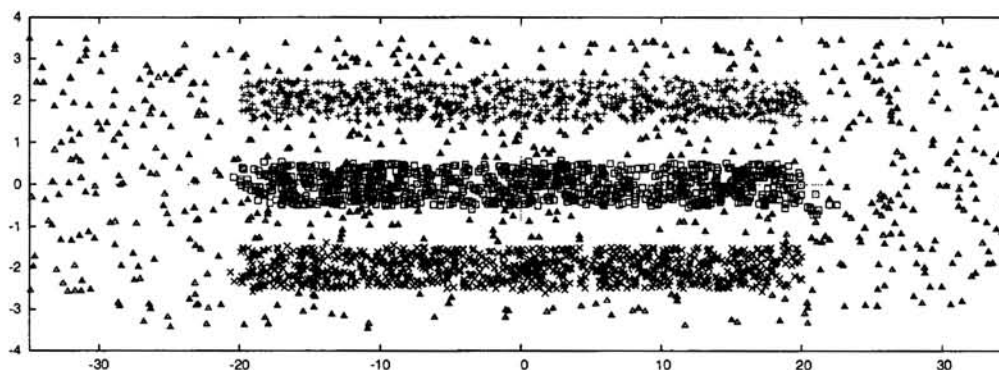

Figure 1: This data set is made of three rectangles, each consisting of 800 points uniformly distributed, and a uniform rectangular background of lower density, also consisting of 800 points. Points classified (with $T_{clus} = 0.08$ and $\theta = 0.5$) as belonging to the three largest clusters are marked by crosses, squares and x's. The fourth cluster is of size 2 and all others are single point clusters marked by triangles.

At high temperatures the system is in a disordered (paramagnetic) phase. As the temperature is lowered, larger and larger regions of high density of points (or spins) exhibit local ordering, until a phase transition occurs and spins in the three rectangular high density regions become completely aligned (*i.e.* within each region all $s_i$ take the same value – super-paramagnetic phase).
The aligned regions define the clusters which we wish to identify. As the temperature

is further lowered, a pseudo-transition occurs and the system becomes completely ordered (ferromagnetic).

# 1   A mean field model

To support our main idea, we analyze an idealized set of points where the division into natural classes is distinct. The points are divided into $M$ groups. The distance between any two points within the same group is $d_1$ while the distance between any two points belonging to different groups is $d_2 > d_1$ ($d$ can be regarded as a similarity index). Following our main idea, we associate a Potts spin with each point and an interaction $J_1$ between points separated by distance $d_1$ and an $J_2$ between points separated by $d_2$, where $0 \leq J_2 < J_1$. Hence the Hamiltonian (1) becomes;

$$\mathcal{H} = -\frac{J_1}{N} \sum_{\mu} \sum_{i<j} \delta_{s_i^\mu, s_j^\mu} - \frac{J_2}{N} \sum_{\mu<\nu} \sum_{i,j} \delta_{s_i^\mu, s_j^\nu} \qquad s_i^\nu = 1, \ldots, q , \qquad (2)$$

where $s_i^\nu$ denotes the $i^{th}$ spin ($i = 1, \ldots, \frac{N}{M}$) of the $\nu^{th}$ group ($\nu = 1, \ldots, M$). From standard mean field theory for the Potts model (Wu, 1982) it is possible to show that the transition from the ferromagnetic phase to the paramagnetic phase is at $T_c = \frac{q-2}{2M(q-1)\log(q-1)} [J_1 + (M-1)J_2]$ . The average spin spin correlation function, $\overline{\delta_{s_i, s_j}}$ at the paramagnetic phase is $\frac{1}{q}$ for all points $\vec{x}_i$ and $\vec{x}_j$; *i.e.* the spin value at each point is independent of the others. The ferromagnetic phase is further divided into two regions. At low temperatures, with high probability, all spins are aligned; that is $\overline{\delta_{s_i, s_j}} \approx 1$ for all $i$ and $j$. At intermediate temperatures, between $T^\star$ and $T_c$, only spins of the same group $\nu$ are aligned with high probability; $\overline{\delta_{s_i^\nu, s_j^\nu}} \approx 1$, while spins belonging to different groups, $\mu$ and $\nu$, are independent; $\overline{\delta_{s_i^\mu, s_j^\nu}} \approx \frac{1}{q}$.

The spin spin correlation function at the super-paramagnetic phase can be used to decide whether or not two spins belong to the same cluster. In contrast with the mere inter-point distance, the spin spin correlation function is sensitive to the collective behavior of the system and is therefore a suitable quantity for defining collective structures (clusters).

The transition temperature $T^\star$ may be calculated and shown to be proportional to $J_2$; $T^\star = \alpha(N, M, q) J_2$. In figure 2 we present the phase diagram, in the $(\frac{T}{J_1}, \frac{J_2}{J_1})$ plane, for the case $M = 4$, $N = 1000$ and $q = 6$.

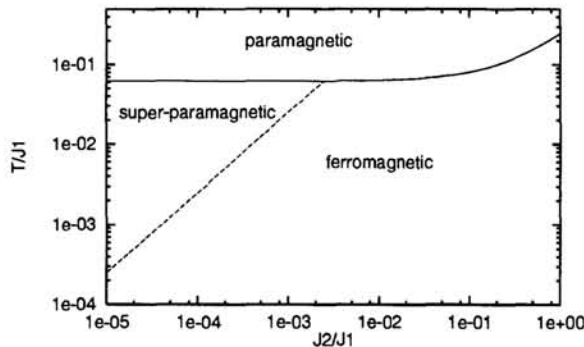

Figure 2: Phase diagram of the mean field Potts model (2) for the case $M = 4$, $N = 1000$ and $q = 6$. The critical temperature $T_c$ is indicated by the solid line, and the transition temperature $T^\star$, by the dashed line.

The phase diagram fig. 2 shows that the existence of natural classes can manifest itself in the thermodynamic properties of the proposed Potts model. Thus our approach is supported, provided that a correct choice of the interaction strengths is made.

## 2   Definition of local interaction

In order to minimize the intra-cluster interaction it is convenient to allow an interaction only between "neighbors". In common with other "local methods", we assume that there is a 'local length scale' $\sim a$, which is defined by the high density regions and is smaller than the typical distance between points in the low density regions. This property can be expressed in the ordering properties of the Potts system by choosing a short range interaction. Therefore we consider that each point interacts only with its neighbors with interaction strength

$$J_{ij} = J_{ji} = \frac{1}{\widehat{K}} \exp\left(-\frac{\|\vec{x}_i - \vec{x}_j\|^2}{2a^2}\right). \tag{3}$$

Two points, $\vec{x}_i$ and $\vec{x}_j$, are defined as neighbors if they have a mutual neighborhood value $K$; that is, if $\vec{x}_i$ is one of the $K$ nearest neighbors of $\vec{x}_j$ and vice-versa. This definition ensures that $J_{ij}$ is symmetric; the number of bonds of any site is less than $K$. We chose the "local length scale", $a$, to be the average of all distances $\|\vec{x}_i - \vec{x}_j\|$ between pairs $i$ and $j$ with a mutual neighborhood value $K$. $\widehat{K}$ is the average number of neighbors per site; *i.e* it is twice the number of non vanishing interactions, $J_{ij}$ divided by the number of points $N$ (This careful normalization of the interaction strength enables us to estimate the critical temperature $T_c$ for any data sample).

## 3   Calculation of thermodynamic quantities

The ordering properties of the system are reflected by the susceptibility and the spin spin correlation function $\overline{\delta_{s_i,s_j}}$, where $\overline{\cdots}$ stands for a thermal average. These quantities can be estimated by averaging over the configurations generated by a Monte Carlo procedure. We use the Swendsen-Wang (Wang and Swendsen, 1990) Monte Carlo algorithm for the Potts model (1) not only because of its high efficiency, but also because it utilizes the SW clusters. As will be explained the SW clusters are strongly connected to the clusters we wish to identify. A layman's explanation of the method is as follows. The SW procedure stochastically identifies clusters of aligned spins, and then flips whole clusters simultaneously. Starting from a given spin configuration, SW go over all the bonds between neighboring points, and either "freeze" or delete them. A bond connecting two neighboring sites $i$ and $j$, is deleted with probability $P_d^{i,j} = \exp(-\frac{J_{ij}}{T}\delta_{s_i,s_j})$ and frozen with probability $P_f^{i,j} = 1 - P_d^{i,j}$. Having gone over all the bonds, all spins which have a path of frozen bonds connecting them are identified as being in the same SW cluster. Note that, according to the definition of $P_d^{i,j}$, only spins of the same value can be frozen in the same SW cluster. Now a new spin configuration is generated by drawing, for each cluster, randomly a value $s = 1, \ldots q$, which is assigned to all its spins. This procedure defines one Monte Carlo step and needs to be iterated in order to obtain thermodynamic averages.

At temperatures where large regions of correlated spins occur, local methods (*e.g.* Metropolis), which flip one spin at a time, become very slow. The SW method overcomes this difficulty by flipping large clusters of aligned spins simultaneously. Hence the SW method exhibits much smaller autocorrelation times than local methods. The strong connection between the SW clusters and the ordering properties of the Potts spins is manifested in the relation

$$\overline{\delta_{s_i,s_j}} = \frac{(q-1)\,\overline{n_{ij}} + 1}{q}, \tag{4}$$

where $n_{ij} = 1$ whenever $s_i$ and $s_j$ belong to the same SW-cluster and $n_{ij} = 0$ otherwise. Thus, $\overline{n_{ij}}$ is the probability that $s_i$ and $s_j$ belong to the same SW-cluster. The r.h.s. of (4) has a smaller variance than its l.h.s., so that the probabilities $\overline{n_{ij}}$ provide an improved estimator of the spin spin correlation function.

## 4  Locating the super-paramagnetic phase

In order to locate the temperature range in which the system is in the super-paramagnetic phase we measure the susceptibility of the system which is proportional to the variance of the magnetization

$$\chi = \frac{N}{T}(\overline{m^2} - \overline{m}^2) \ . \tag{5}$$

The magnetization, $m$, is defined as

$$m = \frac{qN_{\max}/N - 1}{q - 1} \qquad N_{\max} = \max\{N_1, N_2, \ldots N_q\} \ , \tag{6}$$

where $N_\mu$ is the number of spins with the value $\mu$.
In the ferromagnetic phase the fluctuations of the magnetization are negligible, so the susceptibility, $\chi$, is small. As the temperature is raised, a sudden increase of the susceptibility occurs at the transition from the ferromagnetic to the super-paramagnetic phase. The susceptibility is non-vanishing only in the super-paramagnetic phase, which is the only phase where large fluctuations in the magnetization can occur. The point where the susceptibility vanishes again is an upper bound for the transition temperature from the super-paramagnetic to the paramagnetic phase.

## 5  The clustering procedure

Our method consists of two main steps. First we identify the range of temperatures where the clusters may be observed (that corresponding to the super-paramagnetic phase) and choose a temperature within this range. Secondly, the clusters are identified using the information contained in the spin spin correlation function at this temperature. The procedure is summarized here, leaving discussion concerning the choice of the parameters to a later stage.

(a) Assign to each point $\vec{x}_i$ a q-state Potts spin variable $s_i$. q was chosen equal to 20 in the example that we present in this work.

(b) Find all the pairs of points having mutual neighborhood value $K$. We set $K = 10$.

(c) Calculate the strength of the interactions using equation (3).

(d) Use the SW procedure with the Hamiltonian (1) to calculate the susceptibility $\chi$ for various temperatures. The transition temperature from the paramagnetic phase can be roughly estimated by $T_c \approx \frac{e^{-\frac{1}{2}}}{4\log(1+\sqrt{q})}$.

(e) Identify the range of temperatures of non-vanishing $\chi$ (the super-paramagnetic phase). Identify the temperature $T_{max}$ where the susceptibility $\chi$ is maximal, and the temperature $T_{vanish}$, where $\chi$ vanishes at the high temperature side. The optimal temperature to identify the clusters lies between these two temperatures. As a rule of thumb we chose the "clustering temperature" $T_{clus} = \frac{T_{vanish}+T_{max}}{2}$ but the results depend only weakly on $T_{clus}$, as long as $T_{clus}$ is in the super-paramagnetic range, $T_{max} < T_{clus} < T_{vanish}$.

(f) At the clustering temperature $T_{clus}$, estimate the spin spin correlation, $\overline{\delta_{s_i,s_j}}$, for all neighboring pairs of points $\vec{x}_i$ and $\vec{x}_j$, using (4).

(g) Clusters are identified according to a thresholding procedure. The spin spin correlation function $\overline{\delta_{s_i,s_j}}$ of points $\vec{x}_i$ and $\vec{x}_j$ is compared with a threshold, $\theta$; if $\overline{\delta_{s_i,s_j}} > \theta$ they are defined as "friends". Then all mutual friends (including friends of friends, etc) are assigned to the same cluster. We chose $\theta = 0.5$.

In order to show how this algorithm works, let us consider the distribution of points presented in figure 1. Because of the overlap of the larger sparse rectangle with the smaller rectangles, and due to statistical fluctuations, the three dense rectangles actually contain 883, 874 and 863 points.

Going through steps (a) to (d) we obtained the susceptibility as a function of the temperature as presented in figure 3. The susceptibility $\chi$ is maximal at $T_{max} = 0.03$ and vanishes at $T_{vanish} = 0.13$. In figure 1 we present the clusters obtained according to steps (f) and (g) at $T_{clus} = 0.08$. The size of the largest clusters in descending order is 900, 894, 877, 2 and all the rest are composed of only one point. The three biggest clusters correspond to the clusters we are looking for, while the background is decomposed into clusters of size one.

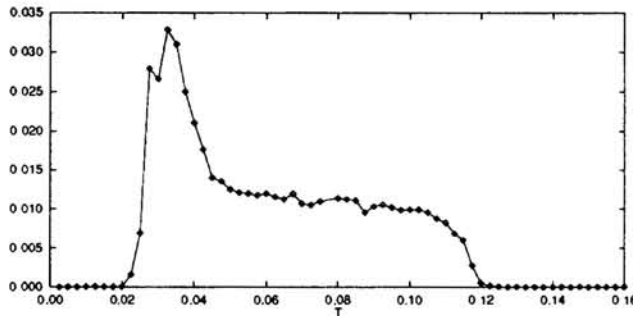

Figure 3: The susceptibility density $\frac{\chi T}{N}$ as a function of the temperature.

Let us discuss the effect of the parameters on the procedure. The number of Potts states, $q$, determines mainly the sharpness of the transition and the critical temperature. The higher $q$, the sharper the transition. On the other hand, it is necessary to perform more statistics (more SW sweeps) as the value of $q$ increases. From our simulations, we conclude that the influence of $q$ is very weak. The maximal number of neighbors, $K$, also affects the results very little; we obtained quite similar results for a wide range of $K$ ($5 \leq K \leq 20$).

No dramatic changes were observed in the classification, when choosing clustering temperatures $T_{clus}$ other than that suggested in (e). However this choice is clearly ad-hoc and a better choice should be found. Our method does not provide a natural way to choose a threshold $\theta$ for the spin spin correlation function. In practice though, the classification is not very sensitive to the value of $\theta$, and values in the range $0.2 < \theta < 0.8$ yield similar results. The reason is that the frequency distribution of the values of the spin spin correlation function exhibits two peaks, one close to $\frac{1}{q}$ and the other close to 1, while for intermediate values it is very close to zero. In figure (4) we present the average size of the largest SW cluster as a function of the temperature, along with the size of the largest cluster obtained by the thresholding procedure (described in (7)) using three different threshold values $\theta = 0.2, 0.5, 0.9$. Note the agreement between the largest cluster size defined by the threshold $\theta = 0.5$ and the average size of the largest SW cluster for all temperatures (This agreement holds for the smaller clusters as well). It supports our thresholding procedure as a sensible one at all temperatures.

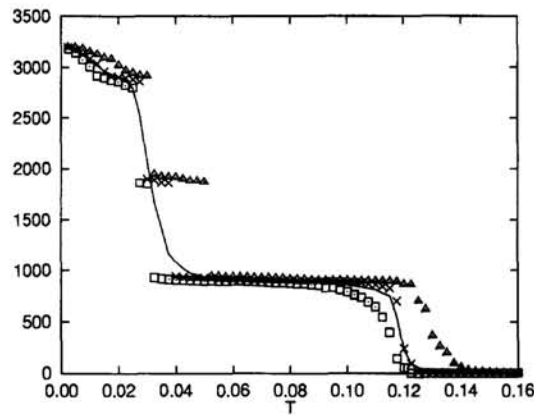

Figure 4: Average size of the largest SW cluster as a function of the temperature, is denoted by the solid line. The triangles, x's and squares denote the size of the largest cluster obtained with thresholds $\theta = 0.2$, 0.5 and 0.9 respectively.

## 6   Discussion

Other methods that were proposed previously, such as Fukunaga's (1990), can be formulated as a Metropolis relaxation of a ferromagnetic Potts model at $T = 0$. The clusters are then determined by the points having the same spin value at the local minima of the energy at which the relaxation process terminates. Clearly this procedure depends strongly on the initial conditions. There is a high probability of getting stuck in a metastable state that does not correspond to the desired answer. Such a $T = 0$ method does not provide any way to distinguish between "good" and "bad" metastable states. We applied Fukunaga's method on the data set of figure (1) using many different initial conditions. The right answer was never obtained. In all runs, domain walls that broke a cluster into two or more parts appeared.

Our method generalizes Fukunaga's method by introducing a finite temperature at which the division into clusters is stable. In addition, the SW dynamics are completely insensitive to the initial conditions and extremely efficient.

Work in progress shows that our method is especially suitable for hierarchical clustering. This is done by identifying clusters at several temperatures which are chosen according to features of the susceptibility curve. In particular our method is successful in dealing with "real life" problems such as the Iris data and Landsat data.

### Acknowledgments

We thank I. Kanter for many useful discussions. This research has been supported by the US-Israel Bi-national Science Foundation (BSF), and the Germany-Israel Science Foundation (GIF).

### References

J.M. Buhmann and H. Kühnel (1993); *Vector quantization with complexity costs*, IEEE Trans. Inf. Theory **39**, 1133.

K. Fukunaga (1990); *Introd. to statistical Pattern Recognition*, Academic Press.

K. Rose, E. Gurewitz, and G.C. Fox (1993); *Constrained clustering as an optimization method*, IEEE Trans on Patt. Anal. and Mach. Intel. **PAMI 15**, 785.

S. Wang and R.H. Swendsen (1990); *Cluster Monte Carlo alg.*, Physica **A 167**, 565.

F.Y. Wu (1982), *The Potts model*, Rev Mod Phys, **54**, 235.

A.L. Yuille and J.J. Kosowsky (1994); *Statistical algorithms that converge*, Neural Computation **6**, 341 (1994).